# Learning invariant features using the Transformed Indian Buffet Process

**Joseph L. Austerweil**
Department of Psychology
University of California, Berkeley
Berkeley, CA 94720
Joseph.Austerweil@gmail.com

**Thomas L. Griffiths**
Department of Psychology
University of California, Berkeley
Berkeley, CA 94720
Tom_Griffiths@berkeley.edu

## Abstract

Identifying the features of objects becomes a challenge when those features can change in their appearance. We introduce the Transformed Indian Buffet Process (tIBP), and use it to define a nonparametric Bayesian model that infers features that can transform across instantiations. We show that this model can identify features that are location invariant by modeling a previous experiment on human feature learning. However, allowing features to transform adds new kinds of ambiguity: Are two parts of an object the same feature with different transformations or two unique features? What transformations can features undergo? We present two new experiments in which we explore how people resolve these questions, showing that the tIBP model demonstrates a similar sensitivity to context to that shown by human learners when determining the invariant aspects of features.

## 1 Introduction

One way the human brain manages the massive amount of sensory information it receives is by learning *invariants* — regularities in its input that do not change across many stimuli sharing some property of interest. Learning and using invariants is essential to many aspects of cognition and perception [1]. For example, the retinal image of an object[1] changes with viewpoint and location, yet people can still identify the object. One explanation for this capability is the visual system recognizes that the features of an object can occur differently across presentations, but will be transformed in a few predictable ways. Representing objects in terms of invariant features poses a challenge for models of feature learning. From a computational perspective, unsupervised feature learning involves recognizing regularities that can be used to compactly encode the observed stimuli [2]. When features have the same appearance and location, techniques such as factorial learning [3] or various extensions of the Indian Buffet Process (IBP) [4] have been successful at learning features, and show some correspondence to human performance [5]. Unfortunately, invariant features do not always have the same appearance or location, by definition. Despite this, people are able to identify invariant features (e.g., [6]), meaning that new machine learning methods need to be explored to fully understand human behavior.

We propose an extension to the IBP called the Transformed Indian Buffet Process (tIBP), which infers features that vary across objects. Analogous to how the Transformed Dirichlet Process extends the Dirichlet Process [7], the tIBP associates a parameter with each instantiation of a feature that determines how the feature is transformed in the given image. This allows for unsupervised learning of features that are invariant in location, size, or orientation. After defining the generative model for the tIBP and presenting a Gibbs sampling inference algorithm, we show that this model can learn visual features that are location invariant by modeling previous behavioral results (from [6]).

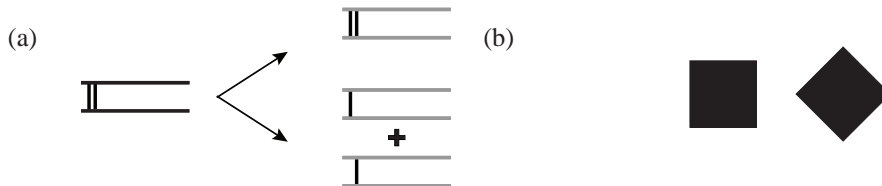

Figure 1: Ambiguous representations. (a) Does this object have one feature that contains two vertical bars or two features that each contain one vertical bar? (b) Are these two shapes the same? The shape on the left is typically perceived as a square and the shape on the right is typically perceived as a diamond despite being objectively equivalent after a transformation (a 45 degree rotation).

One new issue that arises from inferring invariant features is that it can be ambiguous whether parts of an image are the same feature with different transformations or different features. For example, an object containing two vertical bars has (at least) two representations: a single feature containing two vertical bars a fixed distance apart, or two features each of which is a vertical bar with its own translational transformation (see Figure 1 (a)). The tIBP suggests an answer to this question: pick the smallest feature representation that can encode all observed objects. By presenting objects that are either the two vertical bars a fixed distance apart that vary in position or two vertical bars varying independently in location, we confirm that people use sets of objects to infer invariant features in a behavioral experiment and that the different feature representations lead to different decisions.

Introducing transformational invariance also raises the question of what kinds of transformations a feature can undergo. A classic demonstration of the difficulty of defining a set of permissable transformations is the Mach square/diamond [8]. Are the two shapes in Figure 1 (b) the same? The shape on the right is typically perceived as a diamond while the shape on the left is seen as a square, despite being identical except for a rotational transformation. We extend the tIBP to include variables that select the transformations each feature is allowed to undergo. This raises the question of whether people can infer the permissable transformations of a feature. We demonstrate that this is the case by showing that people vary in their generalizations from a square to a diamond depending on whether the square is shown in the context of other squares that vary in rotation. This provides an interesting new explanation of the Mach square/diamond: People learn the allowed transformations of features for a given shape, not what transformations of features are allowed over all shapes.

## 2 Unsupervised feature learning using nonparametric Bayesian statistics

One common approach to unsupervised learning is to explicitly define the generative process that created the observed data. Latent structure can then be identified by inverting this process using Bayesian inference. Nonparametric Bayesian models can be used in this way to infer latent structure of potentially unbounded dimensionality [9]. The Indian Buffet Process (IBP) [4] is a stochastic process that can be used as a prior in nonparametric Bayesian models where each object is represented using an unknown but potentially infinite set of latent features.

### 2.1 Learning features using the Indian Buffet Process

The standard treatment of feature learning using nonparametric Bayesian models factors the observations into two latent structures: (1) a binary matrix $\mathbf{Z}$ that denotes which objects have each feature, and (2) a matrix $\mathbf{Y}$ that represents how the features instantiated. If there are $N$ objects and $K$ features, then $\mathbf{Z}$ is a $N \times K$ binary matrix (where object $n$ has feature $k$ if $z_{nk} = 1$) and $\mathbf{Y}$ is a $K \times D$ matrix (where $D$ is the dimensionality of the observed properties of each object, e.g., the number of pixels in an image). The IBP defines a probability distribution over $\mathbf{Z}$ when $K \to \infty$ such that only a finite number of the columns are non-zero (with prob. 1 for finite $N$). This distribution is

$$P(\mathbf{Z}) = \frac{\alpha^{K_+}}{\prod_{h=1}^{2^N-1} K_h!} \exp\{-\alpha H_N\} \prod_{k=1}^{K_+} \frac{(N - m_k)!(m_k - 1)!}{N!} \tag{1}$$

where $\alpha$ is a parameter affecting the number of non-zero entries in the matrix, $K_h$ is the number of features with history $h$ (the history is the corresponding column of each feature, interpreted as

a binary number), $K_+$ is the number of columns with non-zero entries, $H_N$ is the $N$-th harmonic number, and $m_k$ is the number of objects that have feature $k$. Typically, a simple parametric model is used for $\mathbf{Y}$ (Gaussian for generating real-valued observations, or Bernoulli for binary observations).

The observed properties of objects can be summarized in a $N \times D$ matrix $\mathbf{X}$. The vector $\mathbf{x}_n$ representing the properties of object $n$ is generated based on its features $\mathbf{z}_n$ and the matrix $\mathbf{Y}$. This can be done using a linear-Gaussian likelihood for real-valued properties [4], or a noisy-OR for binary properties [10]. All of the modeling results in this paper use the noisy-OR, with

$$p(x_{nd} = 1|\mathbf{Z}, \mathbf{Y}) \quad = \quad 1 - (1 - \lambda)^{\mathbf{z}_n \mathbf{y}_d}(1 - \epsilon) \tag{2}$$

where $x_{nd}$ is the $d$th observed property of the $n$th object, and $\mathbf{y}_d$ is the corresponding column of $\mathbf{Y}$.

## 2.2 The Transformed Indian Buffet Process (tIBP)

Following Sudderth et al.'s [7] extension of the Dirichlet Process, the Transformed Indian Buffet Process (tIBP) allows features to be transformed. The transformations are object-specific, so in a sense, when an object takes a feature, the feature is transformed with respect to the object. Let $g(\mathbf{Y}|\beta)$ be a prior probability distribution on $\mathbf{Y}$ parameterized by $\beta$, $\Phi(\eta)$ be a distribution over a set of transformations parameterized by $\eta$, $\mathbf{r}_n$ be a vector of transformations of the feature instantiations for object $n$, and $f(\mathbf{x}_n|\mathbf{r}_n(\mathbf{Y}), \mathbf{z}_n, \gamma)$ be the data distribution and $\gamma$ be any other parameters used in the data distribution. The following generative process defines the tIBP:

$$\mathbf{Z}|\alpha \quad \sim \text{IBP}(\alpha) \qquad r_{nk}|\eta \quad \overset{iid}{\sim} \Phi(\eta)$$
$$\mathbf{Y}|\beta \quad \sim g(\beta) \qquad \mathbf{x}_n|\mathbf{r}_n, \mathbf{z}_n, \mathbf{Y}, \gamma \quad \sim f(\mathbf{x}_n|\mathbf{r}_n(\mathbf{Y}), \mathbf{z}_n, \gamma)$$

In this paper, we focus on binary images where the transformations are drawn uniformly at random from a finite set (though Section 5.1 uses a slightly more complicated distribution). The reason for this (instead of using a Dirichlet process over transformations) is that we are interested in modeling invariances in translation, size, or rotation and to model images where a feature occurs in a novel translation, size, or rotation effectively, it is necessary for them to have non-zero probability. In this section, we focus on translations. Assuming our data are in $\{0, 1\}^{D_1 \times D_2}$, a translation shifts the starting place of its feature in each dimension by $r_{nk} = (d_1, d_2)$. We assume a discrete uniform prior on shifts: $r_{nk} \sim U\{0, \ldots, D_1 - 1\} \times U\{0, \ldots, D_2 - 1\}$. Each transformation results in a new interpretation of the feature, $\mathbf{r}_n(\mathbf{y}_d)$. The likelihood $p(x_{nd} = 1|\mathbf{Z}, \mathbf{Y}, \mathbf{R})$ is then identical to Equation 2, substituting the vector of transformed feature interpretations $\mathbf{r}_n(\mathbf{y}_d)$ for $\mathbf{y}_d$.

## 2.3 Inference by Gibbs sampling

We sample from the posterior distribution on feature assignments $\mathbf{Z}$, feature interpretations $\mathbf{Y}$, and transformations $\mathbf{R}$ given observed properties $\mathbf{X}$ using Gibbs sampling [11]. The algorithm consists of iteratively drawing each variable conditioned on the current values of all other variables.

For features with $m_k > 0$ (after removal of the current value of $z_{nk}$), we draw $z_{nk}$ by marginalizing over transformations. This avoids a bottleneck in sampling, as otherwise we would have to get lucky in drawing the right feature and transformation. The marginalization can be done directly, with

$$p(z_{nk}|\mathbf{Z}_{-(nk)}, \mathbf{R}_{-(nk)}, \mathbf{Y}, \mathbf{X}) \quad = \quad \sum_{r_{nk}} p(z_{nk}|\mathbf{Z}_{-(nk)}, \mathbf{R}, \mathbf{Y}, \mathbf{X})p(r_{nk}) \tag{3}$$

where the first term on the right hand side is proportional to $p(\mathbf{x}_n|\mathbf{z}_n, \mathbf{Y}, \mathbf{R})p(z_{nk}|\mathbf{Z}_{-(nk)})$ (provided by the likelihood and the IBP prior respectively, with $\mathbf{Z}_{-(nk)}$ being all of $\mathbf{Z}$ except $z_{nk}$), and the second term is uniform over all $r_{nk}$. If $z_{nk} = 1$, we then sample $r_{nk}$ from

$$p(r_{nk}|z_{nk} = 1, \mathbf{Z}_{-(nk)}, \mathbf{R}_{-(nk)}, \mathbf{Y}, \mathbf{X}) \quad \propto \quad p(\mathbf{x}_n|\mathbf{z}_n, \mathbf{Y}, \mathbf{R})p(r_{nk}) \tag{4}$$

where the relevant probabilities are also used in computing Equation 3, and can thus be cached.

We follow Wood et al.'s [10] method for drawing new features (ie. features for which currently $m_k = 0$). First, we draw an auxilary variable $K_n^{\text{new}}$, the number of "new" features, from

$$p(K_n^{\text{new}}|\mathbf{x}_n, \mathbf{Z}_{n,1:(K+K_n^{\text{new}})}, \mathbf{Y}, \mathbf{R}) \quad \propto \quad p(\mathbf{x}_n|\mathbf{Z}^{\text{new}}, \mathbf{Y}, K_n^{\text{new}})P(K_n^{\text{new}}) \tag{5}$$

where $\mathbf{Z}^{\text{new}}$ is $\mathbf{Z}$ augmented with $K_n^{\text{new}}$ new columns containing ones in row $n$. From the IBP, we know that $K_n^{\text{new}} \sim \text{Poisson}(\alpha/N)$ [4]. To compute the first term on the right hand side, we need to marginalize over the possible new feature images and their transformations ($\mathbf{Y}_{(K+1):(K+K_n^{\text{new}})}$ and $\mathbf{R}_{n,(K+1):(K+K_n^{\text{new}})}$). We assume that the first object to take a feature takes it in its canonical form and thus it is not transformed. Since the first transformation of a feature and its interpretation in an image are not identifiable, this assumption is valid and necessary to aid in inference. With no transformations, drawing the new features in the noisy-OR tIBP model is equivalent to drawing the new features in the normal noisy-OR IBP model. Thus, we can use the same sampling step for $K_n^{\text{new}}$ as [10]. Let $\mathbf{Z}^{\text{new}} = \mathbf{Z}_{n,1:(K+K_n^{\text{new}})}$. Continuing the previous equation,

$$p(K_n^{\text{new}}|\ldots) \quad \propto \quad p(K_n^{\text{new}}) \prod_d p(x_{nd}|\mathbf{Z}^{\text{new}}, \mathbf{Y}, \mathbf{R}, K_n^{\text{new}}) \tag{6}$$

$$= \quad \frac{\alpha^{K_n^{\text{new}}} e^{-\alpha}}{K_n^{\text{new}}!} \left( 1 - (1-\epsilon)(1-\lambda)^{\mathbf{z}_n \mathbf{r}_n(\mathbf{y}_d)} (1-p\lambda)^{K_n^{\text{new}}} \right) \tag{7}$$

where $\mathbf{r}_n(\mathbf{y}_d)$ is the vector of transformed feature interpretations along observed dimension $d$.

Finally, to complete each Gibbs sweep we resample the feature interpretations ($\mathbf{Y}$) given the state of the other variables. We sample each $y_{kd}$ independently given the state of the other variables, with

$$p(y_{kd}|\mathbf{X}, \mathbf{Z}, \mathbf{R}, \mathbf{Y}_{-(kd)}) \quad \propto \quad p(\mathbf{X}|\mathbf{Y}, \mathbf{Z}, \mathbf{R}) p(y_{kd}) \tag{8}$$

where $p(\mathbf{X}|\mathbf{Y}, \mathbf{Z}, \mathbf{R})$ is the likelihood, given by the noisy-OR function.

## 2.4 Prediction

To compare the feature representations our model infers to behavioral results, we need to have judgements of the model for new test objects. This is a prediction problem: computing the probability of a new object $\mathbf{x}_{N+1}$ given the set of $N$ observed objects $\mathbf{X}$. We can express this as

$$P(\mathbf{x}_{N+1}|\mathbf{X}) = \sum_{\mathbf{Z}, \mathbf{Y}, \mathbf{R}} P(\mathbf{x}_{N+1}, \mathbf{Z}, \mathbf{Y}, \mathbf{R}|\mathbf{X}) = \sum_{\mathbf{Z}, \mathbf{Y}, \mathbf{R}} P(\mathbf{x}_{N+1}|\mathbf{Z}, \mathbf{Y}, \mathbf{R}) P(\mathbf{Z}, \mathbf{Y}, \mathbf{R}|\mathbf{X}). \tag{9}$$

The Gibbs sampling algorithm gives us samples from $P(\mathbf{Z}, \mathbf{Y}, \mathbf{R}|\mathbf{X})$ that can be used to approximate this sum. However, a further approximation is required to compute $P(\mathbf{x}_{N+1}|\mathbf{Z}, \mathbf{Y}, \mathbf{R})$. For each sweep of Gibbs sampling, we sample a vector of features $\mathbf{z}_{N+1}$ and corresponding transformations $\mathbf{r}_{N+1}$ for a new object from their conditional distribution given the values of $\mathbf{Z}$, $\mathbf{Y}$, and $\mathbf{R}$ in that sweep, under the constraint that no new features are generated. We use these samples to approximate the calculation of $P(\mathbf{x}_{N+1}|\mathbf{Z}, \mathbf{Y}, \mathbf{R})$ by marginalizing over $\mathbf{z}_{N+1}$ and $\mathbf{r}_{N+1}$.

## 3 Demonstration: Learning Translation Invariant Features

In many situations learners need to form a feature representation of a set of objects, and the features do not reoccur in the exact same location. A common strategy for dealing with this problem is to pre-process data to build in the relevant invariances, or simply to tabulate the presence or absence of features without trying to infer them from the data (e.g., [12]). The tIBP provides a way for a learner to discover that features are translation invariant, and to infer them directly from the data.

Fiser and colleagues [6, 12] showed that when two parts of an image always occur together (forming a "base pair"), people expect the two parts to occur together as if they had one feature representing the pair. In Experiments 1 and 2 of [6], participants viewed 144 scenes, where each scene contained three of the six base pairs in varied spatial location. Each base pair was two of twelve parts in a particular spatial arrangment. Afterwards, participants chose which of two images was more familiar: a base pair (in a never seen before location) and pair of parts that occured together at least once (but were not a base pair). Participants strongly preferred the base pair. To demonstrate the ability of tIBP to infer translation invariant features that are made up of complex parts, we trained the model on the scenes with the same structure as those shown to participants. The only difference was to lower the dimensionality of the images by recoding each part to be a 3 by 3 pixel image (the images from [6] were 1200 by 900 pixels). Figure 2 (a) shows the basic parts (grouped into their base pairs), while 2 (b) shows one scene given to the model. Figure 2 (c) shows the features inferred

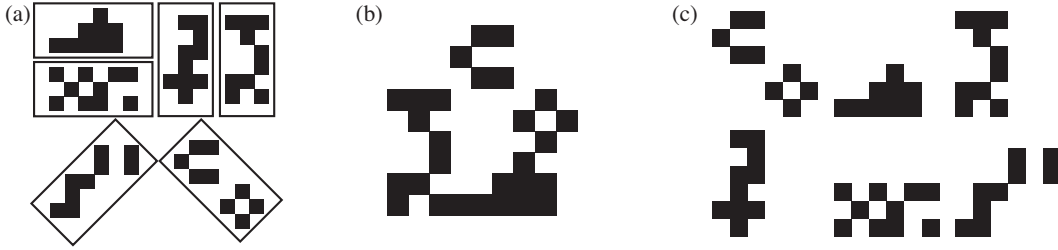

Figure 2: Learning translation invariant features. (a) Each of the parts used to form base pairs, with base pairs grouped in rectangles. (b) One example scene. (c) Features inferred by the tIBP model (one sample from the Gibbs sampler). The tIBP infers the base pairs as features.

by the tIBP model (one sample from the Gibbs sampler after 1000 iterations with a 50 iteration burn-in), given the 144 scenes. The parameters were initialized to $\alpha = 0.8$, $\epsilon = 0.05$, $\lambda = 0.99$, and $p = 0.4$. The model reconstructs the base pairs used to generate the images, and learns that the base pairs can occur in any location. To compare the model people's familiarity judgments, we calculated the model's predictive probability for each base pair in a new location and for a part in that base pair with another part that co-occured with it at least once (but not in a base pair). Over all comparisons, the tIBP model gave higher probability to the image containing the base pair.

## 4 Experiment 1: One feature or two features transformed?

A new problem arises out of learning features that can transform. Is an image composed of the same feature multiple times with different instantiations or is it composed with different features that may or may not be transformed? One way to decide between two possible feature representations for the object is to pick the features that allow you to encode the object and the other objects it is associated with. For example, the object from Figure 1 (a) is the first object (from the top left) in the two sets of objects shown in Figure 3. Figure 3 (a) is the *unitized* object set. All of the objects in this set can be represented as translations of one feature that is two vertical bars. Although this object set can also be described in terms of two features (each of which are vertical bars that can each translate independently), it is a surprising coincidence that the two vertical bars are always the same distance apart over all of the objects in the set. Figure 3 (b) is the *separate* object set. This set is best represented in terms of two features, where each is a vertical bar.

Using different feature representations leads to different predictions about what other objects should be expected to be in the set. Representing the objects with a single feature containing two vertical bars predicts new objects that have vertical bars where the two bars are the same distance apart (*New Unitized*). These objects are also expected under the feature representation that is two features that are each vertical bars; however, any object with two vertical bars is expected (*New Separate*) — not just those with a particular distance apart. Thus, interpreting objects with different feature representations has consequences for how to generalize set membership. In the following experiment, we test these predictions by asking people after viewing either the *unitized* or *separate* object sets to judge how likely the *New Unitized* or *New Separate* objects are to be part of the object set they viewed. We then compare the behavioral results to the features inferred by the tIBP model and the predictive probability of each of the test objects given each of the object sets.

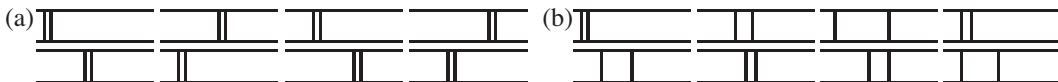

Figure 3: Training sets for Experiment 1. (a) Objects made from spatial translations of the unitized feature. (b) Objects made from spatial translations of two separate features. The number of times each vertical bar is present is the same in the two object sets.

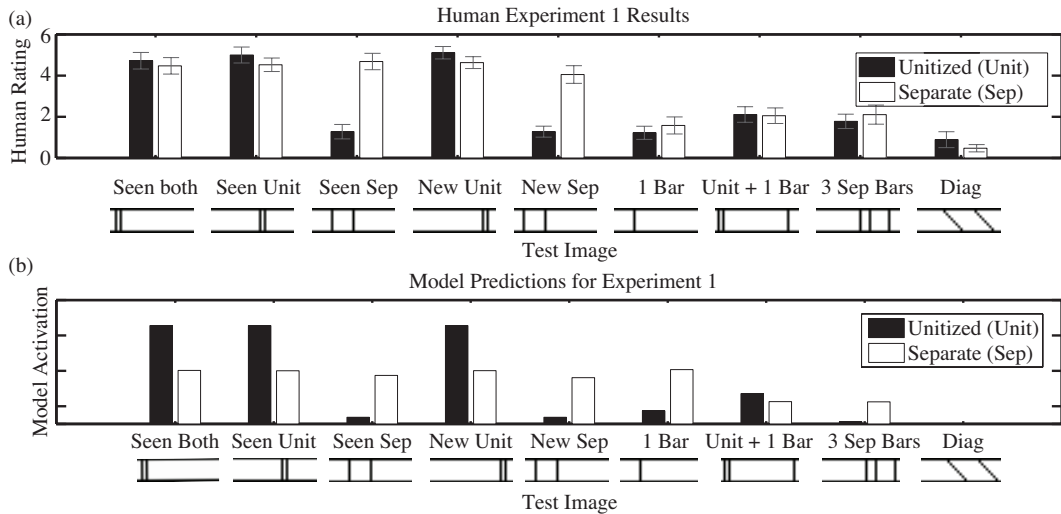

Figure 4: Results of Experiment 1. (a) Human judgments. The *unitized* group only rated those images with two vertical bars close together highly. The *separate* group rate any image with two vertical bars highly. (b) The predictions by the tIBP model.

## 4.1 Methods

A total of 40 participants were recruited online and compensated a small amount. Three participants were removed for failing to complete the task leaving 19 and 18 participants in the *separate* and *unitized* conditions respectively. There were two phases to the experiment: training and test. In the training phase, participants read this cover story (adapted from [13]): "Recently a Mars rover found a cave with a collection of different images on its walls. A team of scientists believes the images could have been left by an alien civilization. The scientists are hoping to understand the images so they can find out about the civilization." They then looked through the eight images (which were either the *unitized* or *separate* object set in a random order) and scrolled down to the next section once they were ready for the test phase. Once they scrolled down to the next section, they were informed that there were many more images on the cave wall that the rover had not yet had a chance to record. Their task for the test phase was to rate how likely on a scale from 0 to 6 they believed the rover would see each image as it explored further through the cave. There were nine test images presented in a random order: *Seen Both* (an image in both training sets), *Seen Unit* (an image that only the *unitized* group saw), *Seen Sep* (an image only the *separate* group saw), *New Unit* (an image valid under the unitized feature set), *New Sep* (a image valid under separate feature set), and four other images that acted as controls (the images are under the horizontal axes of Figure 4).

## 4.2 Results

Figure 4 (a) shows the average ratings made by participants in each group for the nine test images. Over the nine test images, the *separate* group rated the *Seen Sep* ($t(35) = 6.40, p < 0.001$) and *New Sep* ($t(35) = 5.43, p < 0.001$) objects higher than the *unitized* group, but otherwise did not rate any of the other test images significantly different. As predicted by the above analysis, the *unitized* group believed the Mars rover was likely to encounter the two images it observed and the *New Unit* image (the *unitized* feature in a new horizontal position), but did not think it would encounter the other objects. The *separate* group rated any image with two vertical bars highly. This indicates that they represent the images using two features each containing a single vertical bar varying in horizontal position. Thus, each group of participants infer a set of features invariant over the set of observed objects (taking into account the different horizontal position of the features in each object).

Figure 4 (b) shows the predictions made by the tIBP model when given each object set. The predictive probabilities for the test objects were calculated using the procedure outlined above (with the parameter values from Section 3), using 1000 iterations of Gibbs sampling and a 50 iteration burn-in. A non-linear monotonic transformation of these probabilities was used for visualization,

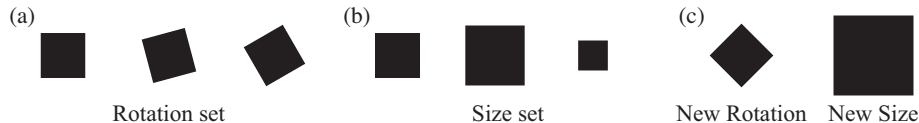

(a)          (b)          (c)

Rotation set          Size set          New Rotation   New Size

Figure 5: Stimuli for investigating how different types of invariances are learned for different object classes. (a) The *rotation* training set. (b) The *size* training set. (c) Two new objects for testing the inferred type of invariance a *New Rotation* and a *New Size* object.

raising the unnormalized probabilities to the power of 0.05 and renormalizing. The Spearman's rank order correlation between the model's predictions and human judgments is 0.85. Qualitatively, the model's predictions are good; however, it incorrectly predicts that the *separate* condition should rate the *1 Bar* test image highly. Unlike the participants in the *separate* condition, the model does not infer that each object has two features and so having only one feature is not a good object. This suggests that while learning the feature representation for a set of objects, people also learn the number of features each object typically has. Investigating how people infer expectations about the number of features objects have is an interesting phenomenon that demands further study.

# 5   Experiment 2: Learning the type of invariance

A natural next step for improving the tIBP would be to make the set of transformations $\Phi$ larger and thus extend the number of possible invariants that can be learned. Although this may be appropriate from a machine learning perspective, it is inappropriate for understanding human cognition. Recall the Mach square/diamond example in Figure 1 (b). Many shapes are equivalent when rotated; however, rotational invariance does not hold for all shapes. This example teaches a counterintuitive moral: The best approach is not to include as many transformations as possible into the model.

Though rotations are not valid transformations for what people commonly consider to be squares, they are appropriate for many objects. This suggests that people infer the set of allowable transformations for different classes of objects. Given the three objects in Figure 5 (a) (the *rotation* set) it seems clear that the *New Rotation* object in Figure 5 (c) belongs in the set, but not the *New Size* object. The reverse holds for the three objects from the left of Figure 5 (b), the *size* set. To explore this phenomenon, we first extend the tIBP to infer the appropriate set of transformations by introducing latent variables for each feature that indicate which transformations it is allowed to use. We demonstrate this extension to the tIBP predicts the *New Rotation* object when given the *rotation* set and predicts the *New Size* object when given the *size* set — effectively learning the appropriate type of invariance for a given object class. Finally, we confirm our introspective argument that people infer the type of invariance appropriate to the observed class of objects.

## 5.1   Learning invariance type using the tIBP

It is straightforward to modify the tIBP such that the type of transformations allowed on a feature is inferred as well. This is done by introducing a hidden variable for each feature that indicate the type of transformation allowed for that feature. Then, the feature transformation is generated conditioned on this hidden variable from a probability distribution specific to the transformation type.

The experiment in this section is learning whether or not the feature defining a set of objects is either rotation or size invariant. Formally, we model this using a generative process that is the same as the tIBP, but introduces the latent variable $t_k$ which determines the type of transformation allowed by feature $k$. If $t_k = 1$, then rotational transformations are drawn from $\Phi_\rho$ (which is the discrete uniform distribution distribution ranging in multiples of fifteen degrees from zero to 45). If $t_k = 0$, then size transformations are drawn from $\Phi_\sigma$ (which is the discrete uniform distribution over $[3/8, 3/7, 3/5, 5/7, 1, 7/5, 11/7, 5/3, 11/5, 7/3, 11/3]$). We assume $t_k \overset{iid}{\sim} \text{Bernoulli}(\pi)$.

The inference algorithm for this extension is the same as for the tIBP except we need to infer the values of $t_k$. We draw $t_k$ using a Gibbs sampling scheme while marginalizing over $r_{1k}, \ldots, r_{nk}$,

$$p(t_k|\mathbf{X}, \mathbf{Y}, \mathbf{Z}, \mathbf{R_{-k}}, \mathbf{t}_{-k}) \quad \propto \quad \sum_{r_{nk}} p(\mathbf{x}_n|r_{nk}, t_k, \mathbf{Y}, \mathbf{Z}, \mathbf{R}_{-k}, \mathbf{t}_{-k})p(\mathbf{r}_k|t_k)p(t_k). \qquad (10)$$

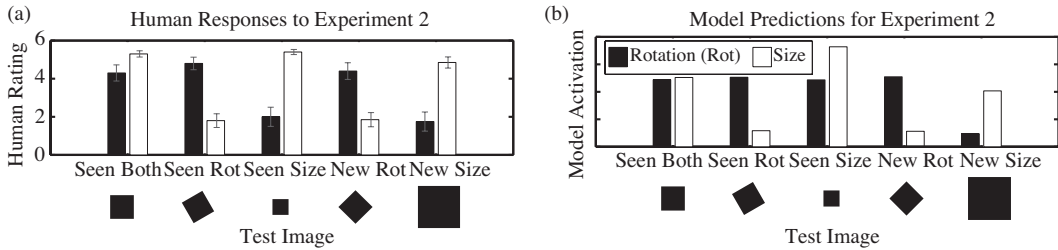

Figure 6: Results of Experiment 2. (a) Responses of human participants. (b) Model predictions.

Prediction is as above except $t_k$ gives the set of transformations each feature is allowed to take.

## 5.2 Methods

A total of 40 participants were recruited online and compensated a small amount, with 20 participants in both training conditions (*rotation* and *size*). The cover story from Experiment 1 was used. Participants observed the three objects in their training set and then generalize on a scale from 0 to 6 to five test objects: *Same Both* (the object that is in both training sets), *Same Rot* (the last object of the *rotation set*), *Same Size* (the last object of the *size set*), *New Rot* and *New Size*.

## 5.3 Results

Figure 6 (a) shows the average human judgments. As expected, participants in the *rotation* condition generalize more to the *New Rot* object than the *size* condition (unpaired $t(38) = 4.44, p < 0.001$) and vice versa for the *New Size* object (unpaired $t(38) = 5.34, p < 0.001$). This confirms our hypothesis; people infer the appropriate set of transformations (a subset of all transformations) features are allowed to use for a class of objects. Figure 6 (b) shows the model predictions with parameters set to $\alpha = 2$, $\epsilon = 0.01$, $\lambda = 0.99$, $p = 0.5$, and $\pi = 0.5$ and using the same visualizing technique as Experiment 1 (with $T = 0.005$), run for 1000 iterations (with a burn-in of 50 iterations) on the sets of images (downsampled to 38 by 38 pixels). Qualitatively, the extended tIBP model has nearly the same pattern of results as the participants in the experiment. The only issue being that it gives high probability to the *Same Size* when given the *rotation set*, an artifact from downsampling. The Spearman's rank order correlation between the model's predictions and human judgments is 0.68. Importantly, the model predicts that only when given the *rotation set* should participants generalize to the *New Rot* object and only when given the *size set* should they generalize to the *New Size* object.

## 6 Conclusions and Future Directions

In this paper, we presented a solution to how people infer feature representations that are invariant over transformations and in two behavioral experiments confirmed two predictions of a new model of human unsupervised feature learning. In addition to these contributions, we proposed a first sketch of a new computational theory of shape representation — the features representing an object are transformed relative to the object and the set of transformations a feature is allowed to undergo depends on the object's context. In the future, we would like to pursue this theory further, expanding the account of learning the types of transformations and exploring how the transformations between features in an object interact (we should expect some interaction due to real world constraints on the transformations, e.g., prospective geometry). Finally, we hope to include other facets of visual perception into our model, like a perceptually realistic prior on feature instantiations and features relations (e.g., the horizontal bar is always ON TOP OF the vertical bar).

**Acknowledgements** We thank Karen Schloss, Stephen Palmer, and the Computational Cognitive Science Lab at Berkeley for discussions and AFOSR grant FA-9550-10-1-0232, and NSF grant IIS-0845410 for support.

## Footnotes

[1]We talk about objects, images, and scenes having features depending on the context.

## References

[1] S. E. Palmer. *Vision Science*. MIT Press, Cambridge, MA, 1999.

[2] H. Barlow. Unsupervised learning. *Neural Computation*, 1:295–311, 1989.

[3] Z. Ghahramani. Factorial learning and the EM algorithm. In *Advances in Neural Information Processing Systems*, volume 7, pages 617–624, Cambridge, MA, 1995. MIT Press.

[4] T. L. Griffiths and Z. Ghahramani. Infinite latent feature models and the Indian buffet process. Technical Report 2005-001, Gatsby Computational Neuroscience Unit, 2005.

[5] J. L. Austerweil and T. L. Griffiths. Analyzing human feature learning as nonparametric Bayesian inference. In Daphne Koller, Yoshua Bengio, Dale Schuurmans, and Léon Bottou, editors, *Advances in Neural Information Processing Systems*, volume 21, Cambridge, MA, 2009. MIT Press.

[6] J. Fiser and R. N. Aslin. Unsupervised statistical learning of higher-order spatial structures from visual scenes. *Psychological Science*, 12(6), 2001.

[7] E. Sudderth, A. Torralba, W. Freeman, and A. Willsky. Describing visual scenes using transformed Dirichlet processes. In *Advances in Neural Information Processing Systems 18*, Cambridge, MA, 2006. MIT Press.

[8] E. Mach. *The analysis of sensations*. Open Court, Chicago, 1914/1959.

[9] M. I. Jordan. Bayesian nonparametric learning: Expressive priors for intelligent systems. In *Heuristics, Probability and Causality: A Tribute to Judea Pearl*. College Publications, 2010.

[10] F. Wood, T. L. Griffiths, and Z. Ghahramani. A non-parametric Bayesian method for inferring hidden causes. In *Proceeding of the 22nd Conference on Uncertainty in Artificial Intelligence*, 2006.

[11] S. Geman and D. Geman. Stochastic relaxation, Gibbs distributions, and the Bayesian restoration of images. *IEEE Transactions on Pattern Analysis and Machine Intelligence*, 6:721–741, 1984.

[12] G. Orban, J. Fiser, R. N. Aslin, and M. Lengyel. Bayesian learning of visual chunks by human observers. *Proceedings of the National Academy of Sciences*, 105(7):2745–2750, 2008.

[13] J. L. Austerweil and T. L. Griffiths. The effect of distributional information on feature learning. In *Proceedings of the Thirty-First Annual Conference of the Cognitive Science Society*. 2009.

